# Multi-dynamic Bayesian Networks

**Karim Filali and Jeff A. Bilmes**
Departments of Computer Science & Engineering and Electrical Engineering
University of Washington
Seattle, WA 98195
{karim@cs,bilmes@ee}.washington.edu

## Abstract

We present a generalization of dynamic Bayesian networks to concisely describe complex probability distributions such as in problems with multiple interacting variable-length streams of random variables. Our framework incorporates recent graphical model constructs to account for existence uncertainty, value-specific independence, aggregation relationships, and local and global constraints, while still retaining a Bayesian network interpretation and efficient inference and learning techniques. We introduce one such general technique, which is an extension of Value Elimination, a backtracking search inference algorithm. *Multi-dynamic Bayesian networks* are motivated by our work on Statistical Machine Translation (MT). We present results on MT word alignment in support of our claim that MDBNs are a promising framework for the rapid prototyping of new MT systems.

## 1 INTRODUCTION

The description of factorization properties of families of probabilities using graphs (i.e., graphical models, or GMs), has proven very useful in modeling a wide variety of statistical and machine learning domains such as expert systems, medical diagnosis, decision making, speech recognition, and natural language processing. There are many different types of graphical model, each with its own properties and benefits, including Bayesian networks, undirected Markov random fields, and factor graphs. Moreover, for different types of scientific modeling, different types of graphs are more or less appropriate. For example, static Bayesian networks are quite useful when the size of set of random variables in the domain does not grow or shrink for all data instances and queries of interest.

Hidden Markov models (HMMs), on the other hand, are such that the number of underlying random variables changes depending on the desired length (which can be a random variable), and HMMs are applicable even without knowing this length as they can be extended indefinitely using online inference. HMMs have been generalized to dynamic Bayesian networks (DBNs) and temporal conditional random fields (CRFs), where an underlying set of variables gets repeated as needed to fill any finite but unbounded length. Probabilistic relational models (PRMs) [5] allow for a more complex template that can be expanded in multiple dimensions simultaneously. An attribute common to all of the above cases is that the specification of rules for expanding any particular instance of a model is finite. In other words, these forms of GM allow the specification of models with an unlimited number of random variables (RVs) using a finite description. This is achieved using parameter tying, so while the number of RVs increases without bound, the number of parameters does not.

In this paper, we introduce a new class of model we call multi-dynamic Bayesian networks. MDBNs are motivated by our research into the application of graphical models to the domain of statistical machine translation (MT) and they have two key attributes from the graphical modeling perspective. First, an MDBN generalizes a DBN in that there are multiple "streams" of variables that can get unrolled, but where each stream may be unrolled by a differing amount. In the most general case, connecting these different streams together would require the specification of conditional probabil-

ity tables with a varying and potentially unlimited number of parents. To avoid this problem and retain the template's finite description length, we utilize a switching parent functionality (also called value-specific independence). Second, in order to capture the notion of fertility in MT-systems (defined later in the text), we employ a form of existence uncertainty [7] (that we call *switching existence*), whereby the existence of a given random variable might depend on the value of other random variables in the network.

Being fully propositional, MDBNs lie between DBNs and PRMs in terms of expressiveness. While PRMs are capable of describing any MDBN, there are, in general, advantages to restricting ourselves to a more specific class of model. For example, in the DBN case, it is possible to provide a bound on inference costs just by looking at attributes of the DBN template only (e.g., the left or right interfaces [12, 2]). Restricting the model can also make it simpler to use in practice. MDBNs are still relatively simple, while at the same time making possible the easy expression of MT systems, and opening doors to novel forms of probabilistic inference as we show below.

In section 2, we introduce MDBNs, and describe their application to machine translation showing how it is possible to represent even complex MT systems. In section 3, we describe MDBN learning and decoding algorithms. In section 4, we present experimental results in the area of statistical machine translation, and future work is discussed in section 5.

## 2  MDBNs

A standard DBN [4] template consists of a directed acyclic graph $\mathcal{G} = (V, E) = (V_1 \cup V_2, E_1 \cup E_2 \cup E_2^{\rightarrow})$ with node set $V$ and edge set $E$. For $t \in \{1, 2\}$, the sets $V_t$ are the nodes at slice $t$, $E_t$ are the intra-slice edges between nodes in $V_t$, and $E_t^{\rightarrow}$ are the inter-slice edges between nodes in $V_1$ and $V_2$. To unroll a DBN to length $T$, the nodes $V_2$ along with the edges adjacent to any node in $V_2$ are cloned $T - 1$ times (where parameters of cloned variables are constrained to be the same as the template) and re-connected at the corresponding places.

An MDBN with $K$ streams consists of the union of $K$ DBN templates along with a template structure specifying rules to connect the various streams together. An MDBN template is a directed graph

$$G = (V, E) = (\bigcup_k V^{(k)}, \bigcup_k E^{(k)} \cup E_{\Updownarrow}^{(k)})$$

where $(V^{(k)}, E^{(k)})$ is the $k^{th}$ DBN, and the edges $E_{\Updownarrow}^{(k)}$ are rules specifying how to connect stream $k$ to the other streams. These rules are general in that they specify the set of edges for all values of $T_k$. There can be arbitrary nesting of the streams such as, for example, it is possible to specify a model that can grow along several dimensions simultaneously.

An MDBN also utilizes "switching existence", meaning some subset of the variables in $V$ bestow existence onto other variables in the network. We call these variables *existence bestowing* (or *eb-nodes*). The idea of bestowing existence is well defined over a discrete space, and is not dissimilar to a variable length DBN. For example, we may have a joint distribution over lengths as follows:

$$p(X_1, \ldots, X_N, N) = p(X_1, \ldots, X_n | N = n)p(N = n)$$

where here $N$ is an eb-node that determines the number of other random variables in the DGM.

Our notion of eb-nodes allows us to model certain characteristics found within machine translation systems, such as "fertility" [3], where a given English word is cloned a random number of times in the generative process that explains a translation from French into English. This random cloning might happen simultaneously at all points along a given MDBN stream. This means that even for a given fixed stream length $T_i = t_i$, each stream could have a randomly varying number of random variables. Our graphical notation for eb-nodes consists of the eb-node as a square box containing variables whose existence is determined by the eb-node.

We start by providing a simple example of an expanded MDBN for three well known MT systems, namely the IBM models 1 and 2 [3], and the "HMM" model [15].[1] We adopt the convention in [3] that our goal is to translate from a string of French words $\mathbf{F} = \mathbf{f}$ of length $M = m$ into a string of English words $\mathbf{E} = \mathbf{e}$ of length $L = l$ — of course these can be any two languages. The basic generative (noisy channel) approach when translating from French to English is to represent the joint

distribution $P(\mathbf{f}, \mathbf{e}) = P(\mathbf{f}|\mathbf{e})P(\mathbf{e})$. $P(\mathbf{e})$ is a language model specifying the prior over the word string $\mathbf{e}$. The key goal is to produce a finite-description length representation for $P(\mathbf{f}|\mathbf{e})$ where $\mathbf{f}$ and $\mathbf{e}$ are of arbitrary length. A hidden alignment string, $\mathbf{a}$, specifies how the English words align to the French word, leading to $P(\mathbf{f}|\mathbf{e}) = \sum_{\mathbf{a}} P(\mathbf{f}, \mathbf{a}|\mathbf{e})$.

Figure 1(a) is a 2-stream MDBN expanded representation of the three models, in this case $\ell = 4$ and $m = 3$. As shown, it appears that the fan-in to node $f_i$ will be $\ell$ and thus will grow without bound. However, a switching mechanism whereby $P(f_i|e, a_i) = P(f_i|e_{a_i})$ limits the number of parameters regardless of $L$. This means that the alignment variable $a_i$ indicates the English word $e_{a_i}$ that should be aligned to French word $f_i$. The variable $e_0$ is a *null* word that connects to French words not explained by any of $e_1, \ldots, e_\ell$. The graph expresses all three models — the difference is that, in Models 1 and 2, there are no edges between $a_j$ and $a_{j+1}$. In Model 1, $p(a_j = \ell)$ is uniform on the set $\{1, \ldots, L\}$; in Model 2, the distribution over $a_j$ is a function only of its position $j$, and on the English and French lengths $\ell$ and $m$ respectively. In the M-HMM model, the $a_i$ variables form a first order Markov chain.

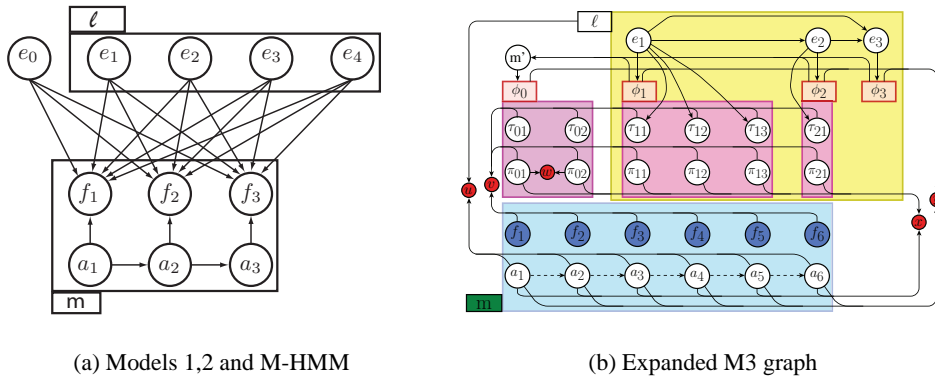

(a) Models 1,2 and M-HMM        (b) Expanded M3 graph

Figure 1: *Expanded 2-stream MDBN description of IBM Models 1 and 2, and the M-HMM model for MT; and the expanded MDBN description of IBM Model 3 with fertility assignment $\phi_0 = 2, \phi_1 = 3, \phi_2 = 1, \phi_3 = 0$.*

From the above, we see that it would be difficult to express this model graphically using a standard DBN since $L$ and $M$ are unequal random variables. Indeed, there are two DBNs in operation, one consisting of the English string, and the other consisting of the French string and its alignment. Moreover, the fully connected structure of the graph in the figure can represent the appropriate family of model, but it also represents models whose parameter space grows without bound — the switching function allows the model template to stay finite regardless of $L$ and $M$.

With our MDBN descriptive abilities complete, it is now possible to describe the more complex IBM models 3, and 4[3] (an MDBN for Model3 is depicted in fig. 1(b)). The top most random variable, $\ell$, is a hidden switching existence variable corresponding to the length of the English string. The box abutting $\ell$ includes all the nodes whose existence depends on the value of $\ell$. In the figure, $\ell = 3$, thus resulting in three English words $e_1, e_2$, and $e_3$ connected using a second-order Markov chain. To each English word $e_i$ corresponds a conditionally dependent fertility eb-node $\phi_i$, which indicates how many times $e_i$ is used by words in the French string. Each $\phi_i$ in turn controls the existence of a set of variables under it. Given the fertilities (the figure depicts the case $\phi_1 = 3, \phi_2 = 1, \phi_3 = 0$), for each word $e_i$, $\phi_i$ French word variables are granted existence and are denoted by $\tau_{i1}, \tau_{i2}, \ldots, \tau_{i\phi_i}$, what is called the *tablet* [3] of $e_i$. The values taken by the $\tau$ variables need to match the actual observed French sequence $f_1, \ldots, f_m$. This is represented as a shared constraint between all the $f$, $\pi$, and $\tau$ variables which have incoming edges into the observed variable $v$. $v$'s conditional probability table is such that it is one only when the associated constraint is satisfied[2]. The variable

$\pi_{i,k} \in \{1, \ldots, m\}$ is a switching dependency parent with respect to the constraint variable $v$ and determines which $f_j$ participates in an equality constraint with $\tau_{i,k}$.

The bottom variable $m$ is a switching existence node (observed to be 6 in the figure) with corresponding French word sequence and alignment variables. The French sequence participates in the $v$ constraint described above, while the alignment variables $a_j \in \{1, \ldots, \ell\}, j \in 1, \ldots, m$ constrain the fertilities to take their unique allowable values (for the given alignment). Alignments also restrict the domain of permutation variables, $\boldsymbol{\pi}$, using the constraint variable $x$. Finally, the domain size of each $a_j$ has to lie in the interval $[0, \ell]$ and that is enforced by the variable $u$. The dashed edges connecting the alignment $a$ variables represent an extension to implement an M3/M-HMM hybrid.

The *null submodel* involving the deterministic node $m'(= \sum_{i=1}^{\ell} \phi_i)$ and eb-node $\phi_0$ accounts for French words that are not explained by any of the English words $e_1, \ldots, e_\ell$. In this submodel, successive permutation variables are ordered and this constraint is implemented using the observed child $w$ of $\pi_{0i}$ and $\pi_{0(i+1)}$.

Model 4 [3] is similar to Model 3 except that the former is based on a more elaborate distortion model that uses relative instead of absolute positions both within and between tablets.

## 3 Inference, Parameter Estimation and MPE

Multi-dynamic Bayesian Networks are amenable to any type of inference that is applicable to regular Bayesian networks as long as switching existence relationships are respected and all the constraints (aggregation for example) are satisfied. Unfortunately DBN inference procedures that take advantage of the repeatable template and can preprocess it offline, are not easy to apply to MDBNs. A case in point is the Junction Tree algorithm [11]. Triangulation algorithms exist that create an offline triangulated version of the input graph and do not re-triangulate it for each different instance of the input data [12, 2]. In MDBNs, due to the flexibility to unroll templates in several dimensions and to specify dependencies and constraints spanning the entire unrolled graph, it is not obvious how we can exploit any repetitive patterns in a Junction Tree-style offline triangulation of the graph template.

In section 4, we discuss sampling inference methods we have used. Here we discuss our extension to a backtracking search algorithm with the same performance guarantees as the JT algorithm, but with the advantage of easily handling determinism, existence uncertainty, and constraints, both learned and explicitly stated.

Value Elimination (VE) ([1]), is a backtracking Bayesian network inference technique that caches **factors** associated with portions of the search tree and uses them to avoid iterating again over the same subtrees. We follow the notation introduced in [1] and refer the reader to that paper for details about VE inference. We have extended the VE inference approach to handle explicitly encoded constraints, existence uncertainty, and to perform *approximate local domain pruning* (see section 4). We omit these details as well as others in the original paper and briefly describe the main data structure required by VE and sketch the algorithm we refer to as **FirstPass** (fig. 1) since it constitutes the first step of the learning procedure, our main contribution in this section.

A VE **factor**, $F$, is such that we can write the following marginal of the joint distribution

$$\sum_{\mathbf{X}=\mathbf{x}} P(\mathbf{X} = \mathbf{x}, \mathbf{Y} = \mathbf{y}, \mathbf{Z}) = F.val \times f(\mathbf{Z})$$

such that $(\mathbf{X} \cup \mathbf{Y}) \cap \mathbf{Z} = \emptyset$, $F.val$ is a constant, and $f(\mathbf{Z})$ a function of $\mathbf{Z}$ only. $\mathbf{Y}$ is a set of variables previously instantiated in the current branch of search tree to the value vector $\mathbf{y}$. The pair $(\mathbf{Y}, \mathbf{y})$ is referred to as a **dependency set (F.Dset)**. $\mathbf{X}$ is referred to as a **subsumed set (F.Sset)**. By caching the tuple $(F.Dset, F.Sset, F.val)$, we avoid recomputing the marginal again whenever (1) $F.Dset$ is **active**, meaning all nodes stored in $F.Dset$ are assigned their cached values in the current branch of the search tree; and (2) none of the variables in $F.Sset$ are assigned yet.

**FirstPass** (alg. 1) visits nodes in the graph in Depth First fashion. In line 7, we get the values of all **Newly Single-valued (NSV)** CPTs i.e., CPTs that involve the current node, $V$, and in which all

---

We use a general directed domain pruning constraint. Deterministic relationships then become a special case of our constraint whereby the domain of the child variable is constrained to a single value with probability one.

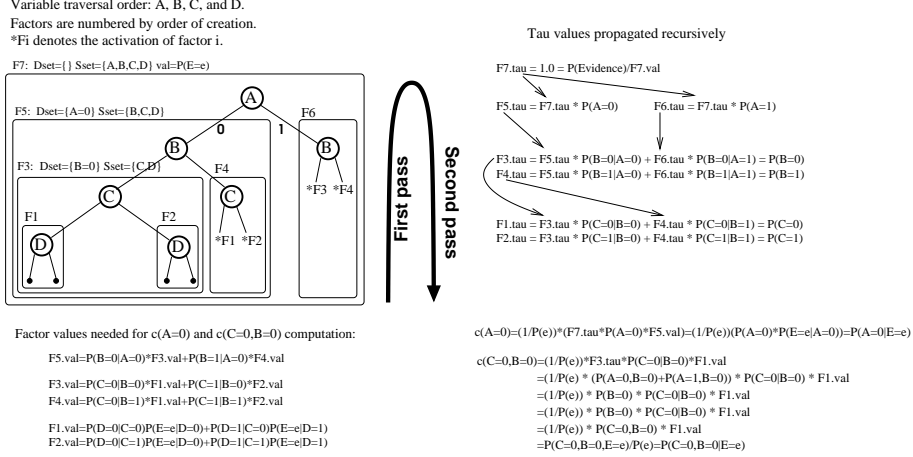

Figure 2: *Learning example using the Markov chain $A \rightarrow B \rightarrow C \rightarrow D \rightarrow E$, where $E$ is observed. In the first pass, factors (Dset, Sset and val) are learned in a bottom up fashion. Also, the normalization constant $P(E = e)$ (probability of evidence) is obtained. In the second pass, tau values are updated in a top-down fashion and used to calculate expected counts $c(F.head, pa(F.head))$ corresponding to each F.head (the figure shows the derivations for (A=0) and (C=0,B=0), but all counts are updated in the same pass).*

other variables are already assigned (these variables and their values are accumulated into Dset). We also check for factors that are active, multiply their values in, and accumulate subsumed vars in Sset (to avoid branching on them). In line 10, we add $V$ to the Sset. In line 11, we cache a new factor $F$ with value $F.val = sum$. We store $V$ into **F.head**, a pointer to the last variable to be inserted into $F.Sset$, and needed for parameter estimation described below. $F.Dset$ consists of all the variables, except $V$, that appeared in any NSV CPT or the Dset of an activated factor at line 6.

Regular Value Elimination is query-based, similar to variable elimination and recursive conditioning—what this means is that to answer a query of the type $P(Q|\mathbf{E} = \mathbf{e})$, where $Q$ is query variable and $\mathbf{E}$ a set of evidence nodes, we force $Q$ to be at the top of the search tree, run the backtracking algorithm and then read the answers to the queries $P(Q = q|\mathbf{E} = \mathbf{e}), q \in Dom[Q]$, along each of the outgoing edges of $Q$. Parameter estimation would require running a number of queries on the order of the number of parameters to estimate.

We extend VE into an algorithm that allows us to obtain Expectation Maximization sufficient statistics in a single run of Value Elimination plus a second pass, which can never take longer than the first one (and in practice is much faster). This two-pass procedure is analogous to the collect-distribute evidence procedure in the Junction Tree algorithm, but here we do this via a search tree.

Let $\theta_{X=x|\mathbf{pa}(\mathbf{X})=\mathbf{y}}$ be a parameter associated with variable $X$ with value $x$ and parents $\mathbf{Y} = \mathbf{pa}(\mathbf{X})$ when they have value $\mathbf{y}$. Assuming a maximum likelihood learning scenario[3], to estimate $\theta_{X=x|\mathbf{pa}(\mathbf{X})=\mathbf{y}}$, we need to compute

$$f(X = x, \mathbf{pa}(\mathbf{X}) = \mathbf{y}, \mathbf{E} = \mathbf{e}) = \sum_{\mathbf{W}\backslash\{X,\mathbf{pa}(\mathbf{X})\}} P(\mathbf{W}, X = x, \mathbf{pa}(\mathbf{X}) = \mathbf{y}, \mathbf{E} = \mathbf{e})$$

which is a sum of joint probabilities of all configurations that are consistent with the assignment $\{X = x, \mathbf{pa}(\mathbf{X}) = \mathbf{y}\}$. If we were to turn off factor caching, we would enumerate all such variable configurations and could compute the sum. When standard VE factors are used, however, this is no longer possible whenever $X$ or any of its parents becomes subsumed. Fig. 2 illustrates an example of a VE tree and the factors that are learned in the case of a Markov chain with an evidence node at the end. We can readily estimate the parameters associated with variables $A$ and $B$ as they are not subsumed along any branch. $C$ and $D$ become subsumed, however, and we cannot obtain the correct counts along all the branches that would lead to $C$ and $D$ in the full enumeration case.

To address this issue, we store a special value, $F.tau$, in each factor. $F.tau$ holds the sum over all path probabilities from the first level of the search tree to the level at which the factor $F$ was

either created or activated. For example, $F6.tau$ in fig. 2 is simply $P(A = 1)$. Although we can compute $F3.tau$ directly, we can also compute it recursively using $F5.tau$ and $F6.tau$ as shown in the figure. This is because both $F5$ and $F6$ *subsume* $F3$: in the context $\{F5.Dset\}$, there exists a (unique) value $d_{sub}$ of $F5.head$[4] s.t. $F3$ becomes activable. Likewise for $F6$. We cannot compute $F1.tau$ directly, but we can, recursively, from $F3.tau$ and $F4.tau$ by taking advantage of a similar subsumption relationship. In general, we can show that the following recursive relationship holds:

$$F.tau \leftarrow \sum_{F^{pa} \in \mathcal{F}^{pa}} F^{pa}.tau \times NSV_{F^{pa}.head=d_{sub}} \times \frac{\prod_{F_{act} \in \mathcal{F}_{act}} F_{act}.val}{F.val} \qquad (1)$$

where $\mathcal{F}^{pa}$ is the set of factors that subsume $F$, $\mathcal{F}_{act}$ is the set of all factors (including $F$) that become active in the context of $\{F^{pa}.Dset, F^{pa}.head = d_{sub}\}$ and $NSV_{F^{pa}.head=d_{sub}}$ is the product of all newly single valued CPTs under the same context. For top-level factors (not subsumed by any factor), $F.tau = P_{evidence}/F.val$, which is 1.0 when there is a unique top-level factor.

Alg. 2 is a simple recursive computation of eq. 1 for each factor. We visit learned factors in the reverse order in which they were learned to ensure that, for any factor $F'$, $F'.tau$ is incremented (line 13) by any $F$ that might have activated $F'$ (line 12). For example, in fig. 2, $F4$ uses $F1$ and $F2$, so $F4.tau$ needs to be updated before $F1.tau$ and $F2.tau$. In line 11, we can increment the counts for any NSV CPT entries since $F.tau$ will account for the possible ways of reaching the configuration $\{F.Dset, F.head = d\}$ in an equivalent full enumeration tree.

---

**Algorithm 1**: FirstPass(level)

**Input**: Graph $G$
**Output**: A list of learned factors and $P_{evidence}$
1 Select var $V$ to branch on
2 **if** $V == NONE$ **then** return
3 Sset={}, Dset={}
4 **for** $d \in Dom[V]$ **do**
5      $V \leftarrow d$
6      $prod$ = productOfAllNSVsAndActiveFactors(Dset, Sset)
7      **if** $prod \mathrel{!}= 0$ **then** FirstPass(level+1)
8      sum += prod
9 $Sset = Sset \cup \{V\}$
10 cacheNewFactor($F.head \leftarrow V$, $F.val \leftarrow sum$, $F.Sset \leftarrow Sset$, $F.Dset \leftarrow Dset$);

---

**Algorithm 2**: SecondPass()

**Input**: $\mathcal{F}$: List of factors in the reverse order learned in the first pass and $P_{evidence}$.
**Result**: Updated counts
1 **foreach** $F \in \mathcal{F}$ **do**
2      **if** $F.Dset = \{\}$ **then**
3          $F.tau \leftarrow P_{evidence}/F.val$
4      **else**
5          $F.tau \leftarrow 0.0$
6          Assign vars in $F.Dset$ to their values
7      $V \leftarrow F.head$ (last node to have been subsumed in this factor)
8      **foreach** $d \in Dom[V]$ **do**
9          $prod$ = productOfAllNSVsAndActiveFactors()
10          $prod* = F.tau$
11          **foreach** *newly single-valued CPT $C$* **do** count(C.child,C.parents)+=$prod/P_{evidence}$
12          $\mathcal{F}'$=getListOfActiveFactors()
13          **for** $F' \in \mathcal{F}'$ **do** $F'.tau + = prod/F'.val$

---

**Most Probable Explanation**    We compute MPE using a very similar two-pass algorithm. In the first pass, factors are used to store a maximum instead of a summation over variables in the Sset. We also keep track of the value of $F.head$ at which the maximum is achieved. In the second pass, we recursively find the optimal variable configuration by following the trail of factors that are activated when we assign each $F.head$ variable to its maximum value starting from the last learned factor.

# 4 MACHINE TRANSLATION WORD ALIGNMENT EXPERIMENTS

A major motivation for pursuing the type of representation and inference described above is to make it possible to solve computationally-intensive real-world problems using large amounts of data, while retaining the full generality and expressiveness afforded by the MDBN modeling language. In the experiments below we compare running times of MDBNs to GIZA++ on IBM Models 1 through 4 and the M-HMM model. GIZA++ is a special-purpose optimized MT word alignment C++ tool that is widely used in current state-of-the-art phrase-based MT systems [10] and at the time of this writing is the only publicly available software that implements all of the IBM Models. We test on French-English 107 hand-aligned sentences[5] from a corpus of the European parliament proceedings (Europarl [9]) and train on 10000 sentence pairs from the same corpus and of maximum number of words 40. The Alignment Error Rate (AER) [13] evaluation metric quantifies how well the MPE assignment to the hidden alignment variables matches human-generated alignments.

Several pruning and smoothing techniques are used by GIZA and MDBNs. GIZA prunes low lexical ($P(f|e)$) probability values and uses a default small value for unseen (or pruned) probability table entries. For models 3 and 4, for which there is no known polynomial time algorithm to perform the full E-step or compute MPE, GIZA generates a set of high probability alignments using an M-HMM and hill-climbing and collects EM counts over these alignments using M3 or M4. For MDBN models we use the following pruning strategy: at each level of the search tree we prune values which, together, account for the lowest specified percentage of the total probability mass of the product of all newly active CPTs in line 6 of alg. 1. This is a more effective pruning than simply removing low-probability values of each CPD because it factors in the joint contribution of multiple active variables.

Table 1 shows a comparison of timing numbers obtained GIZA++ and MDBNs. The runtime numbers shown are for the combined tasks of training and decoding; however, training time dominates given the difference in size between train and test sets. For models 1 and 2 neither GIZA nor MDBNs perform any pruning. For the M-HMM, we prune 60% of probability mass at each level and use a Dirichlet prior over the alignment variables such that long-range transitions are exponentially less likely than shorter ones.[6] This model achieves similar times and AER to GIZA's. Interestingly, without any pruning, the MDBN M-HMM takes 160 minutes to complete while only marginally improving upon the pruned model. Experimenting with several pruning thresholds, we found that AER would worsen much more slowly than runtime decreases.

Models 3 and 4 have treewidth equal to the number of alignment variables (because of the global constraints tying them) and therefore require approximate inference. Using Model 3, and a drastic pruning threshold that only keeps the value with the top probability at each level, we were able to achieve an AER not much higher than GIZA's. For M4, it achieves a best AER of 31.7% while we do not improve upon Model3, most likely because a too restrictive pruning. Nevertheless, a simple variation on Model3 in the MDBN framework achieves a lower AER than our regular M3 (with pruning still the same). The M3-HMM hybrid model combines the Markov alignment dependencies from the M-HMM model with the fertility model of M3.

**MCMC Inference**   Sampling is widely used for inference in high-treewidth models. Although MDBNs support Likelihood Weighing, it is very inefficient when the probability of evidence is very small, as is the case in our MT models. Besides being slow, Markov chain Monte Carlo can be problematic when the joint distribution is not positive everywhere, in particular in the presence of determinism and hard constraints. Techniques such as blocking Gibbs sampling [8] try to address the problem. Often, however, one has to carefully choose a problem-dependent proposal distribution. We used MCMC to improve training of the M3-HMM model. We were able to achieve an AER of 32.8% (down from 39.1%) but using 400 minutes of uniprocessor time.

# 5 CONCLUSION

The existing classes of graphical models are not ideally suited for representing SMT models because "natural" semantics for specifying the latter combine flavors of different GM types on top of standard directed Bayesian network semantics: *switching parents* found in Bayesian Multinets [6], aggregation relationships such as in Probabilistic Relational Models [5], and existence uncertainty [7]. We

| Model | GIZA++ | | MDBN | |
|---|---|---|---|---|
| Init | **M1** | **M-HMM** | **M1** | **M-HMM** |
| **M1** | 1m45s (47.7%) | N/A | 3m20s (48.0%) | N/A |
| **M2** | 2m02s (41.3%) | N/A | 5m30s (41.0%) | N/A |
| **M-HMM** | 4m05s (35.0%) | N/A | 4m15s (33.0%) | N/A |
| **M3** | 2m50s (45%) | 5m20s (38.5%) | 12m (43.6%) | 9m (42.5%) |
| **M4** | 5m20s (34.8%) | 7m45s (31.7%) | 25m (43.6%) | 23m (42.6%) |
| **M3-HMM** | N/A | | 9m30 (41.0%) | 9m15s (39.1%) MCMC 400m (32.8%) |

Table 1: *MDBN VE-based learning versus GIZA++ timings and %AER using 5 EM iterations. The columns M1 and M-HMM correspond to the model that is used to initialize the model in the corresponding row. The last row is a hybrid Model3-HMM model that we implemented using MDBNs and is not expressible using GIZA.*

have introduced a generalization of dynamic Bayesian networks to easily and concisely build models consisting of varying-length parallel asynchronous and interacting data streams. We have shown that our framework is useful for expressing various statistical machine translation models. We have also introduced new parameter estimation and decoding algorithms using exact and approximate search-based probability computation. While our timing results are not yet as fast as a hand-optimized C++ program on the equivalent model, we have shown that even in this general-purpose framework of MDBNs, our timing numbers are competitive and usable. Our framework can of course do much more than the IBM and HMM models. One of our goals is to use this framework to rapidly prototype novel MT systems and develop methods to statistically induce an interlingua. We also intend to use MDBNs in other domains such as multi-party social interaction analysis.

## Footnotes

[1]We will refer to it as M-HMM to avoid confusion with regular HMMs.

[2]This type of encoding of constraints corresponds to the standard mechanism used by Pearl [14]. A naive implementation, however, would enumerate a number of configurations exponential in the number of constrained variables, while typically only a small fraction of the configurations would have positive probability.

[3]For Bayesian networks the likelihood function decomposes such that maximizing the expectation of the complete likelihood is equivalent to maximizing the "local likelihood" of each variable in the network.

[4]Recall, $F.head$ is the last variable to be added to a newly created factor in line 10 of alg. 1

[5]Available at `http://www.cs.washington.edu/homes/karim`

[6]French and English have similar word orders. On a different language pair, a different prior might be more appropriate. With a uniform prior, the MDBN M-HMM has 36.0% AER.

## References

[1] F. Bacchus, S. Dalmao, and T. Pitassi. Value elimination: Bayesian inference via backtracking search. In *UAI-03*, pages 20–28, San Francisco, CA, 2003. Morgan Kaufmann.

[2] J. Bilmes and C. Bartels. On triangulating dynamic graphical models. In *Uncertainty in Artificial Intelligence: Proceedings of the 19th Conference*, pages 47–56. Morgan Kaufmann, 2003.

[3] P. F. Brown, J. Cocke, S. A. Della Piettra, V. J. Della Piettra, F. Jelinek, J. D. Lafferty, R. L. Mercer, and P. S. Roossin. A statistical approach to machine translation. *Computational Linguistics*, 16(2):79–85, June 1990.

[4] T. Dean and K. Kanazawa. Probabilistic temporal reasoning. *AAAI*, pages 524–528, 1988.

[5] N. Friedman, L. Getoor, D. Koller, and A. Pfeffer. Learning probabilistic relational models. In *IJCAI*, pages 1300–1309, 1999.

[6] D. Geiger and D. Heckerman. Knowledge representation and inference in similarity networks and Bayesian multinets. *Artif. Intell.*, 82(1-2):45–74, 1996.

[7] L. Getoor, N. Friedman, D. Koller, and B. Taskar. Learning probabilistic models of link structure. *Journal of Machine Learning Research*, 3(4-5):697–707, May 2003.

[8] C. Jensen, A. Kong, and U. Kjaerulff. Blocking Gibbs sampling in very large probabilistic expert systems. In *International Journal of Human Computer Studies. Special Issue on Real-World Applications of Uncertain Reasoning.*, 1995.

[9] P. Koehn. Europarl: A multilingual corpus for evaluation of machine translation. http://www.isi.edu/koehn/publications/europarl, 2002.

[10] P. Koehn, F. Och, and D. Marcu. Statistical phrase-based translation. In *NAACL/HLT 2003*, 2003.

[11] S. Lauritzen. *Graphical Models*. Oxford Science Publications, 1996.

[12] K. Murphy. *Dynamic Bayesian Networks: Representation, Inference and Learning*. PhD thesis, U.C. Berkeley, Dept. of EECS, CS Division, 2002.

[13] F. J. Och and H. Ney. Improved statistical alignment models. In *ACL*, pages 440–447, Oct 2000.

[14] J. Pearl. *Probabilistic Reasoning in Intelligent Systems: Networks of Plausible Inference*. Morgan Kaufmann, 2nd printing edition, 1988.

[15] S. Vogel, H. Ney, and C. Tillmann. HMM-based word alignment in statistical translation. In *Proceedings of the 16th conference on Computational linguistics*, pages 836–841, Morristown, NJ, USA, 1996.
